# QUIC-SVD: Fast SVD Using Cosine Trees

**Michael P. Holmes, Alexander G. Gray and Charles Lee Isbell, Jr.**
College of Computing
Georgia Tech
Atlanta, GA 30327
{mph, agray, isbell}@cc.gatech.edu

## Abstract

The Singular Value Decomposition is a key operation in many machine learning methods. Its computational cost, however, makes it unscalable and impractical for applications involving large datasets or real-time responsiveness, which are becoming increasingly common. We present a new method, QUIC-SVD, for fast approximation of the whole-matrix SVD based on a new sampling mechanism called the *cosine tree*. Our empirical tests show speedups of several orders of magnitude over exact SVD. Such scalability should enable QUIC-SVD to accelerate and enable a wide array of SVD-based methods and applications.

## 1 Introduction

The Singular Value Decomposition (SVD) is a fundamental linear algebraic operation whose abundant useful properties have placed it at the computational center of many methods in machine learning and related fields. Principal component analysis (PCA) and its kernel and nonlinear variants are prominent examples, and countless other instances are found in manifold and metric learning, clustering, natural language processing/search, collaborative filtering, bioinformatics and more.

Notwithstanding the utility of the SVD, it is critically bottlenecked by a computational complexity that renders it impractical on massive datasets. Yet massive datasets are increasingly common in applications, many of which require real-time responsiveness. Such applications could use SVD-based methods more liberally if the SVD were not so slow to compute. We present a new method, QUIC-SVD, for fast, sample-based SVD approximation with automatic relative error control. This algorithm is based on a new type of data partitioning tree, the cosine tree, that shows excellent ability to home in on the subspaces needed for good SVD approximation. We demonstrate several-order-of-magnitude speedups on medium-sized datasets, and verify that approximation error is properly controlled. Based on these results, QUIC-SVD seems able to help address the scale of modern problems and datasets, with the potential to benefit a wide array of methods and applications.

## 2 Background

For $A \in \mathbb{R}^{m \times n}$, we write $A_{(i)}$ for the $i$th row of $A$ and $A^{(j)}$ for the $j$th column. We use $\mathbb{O}^{m \times n}$ to represent the subset of $\mathbb{R}^{m \times n}$ whose columns are orthonormal. Since the columns of $V \in \mathbb{O}^{m \times n}$ are an orthonormal basis, we sometimes use expressions such as "the subspace $V$" to refer to the subspace *spanned* by the columns of $V$. Throughout this paper we assume $m \geq n$, such that sampling rows gives bigger speedup than sampling columns. This is no loss of generality, since whenever $m < n$ we can perform SVD on the transpose, then swap $U$ and $V$ to get the SVD of the original matrix. Alternatively, row-sampling-based methods have analogous column-sampling versions that can be used in place of transposition; we leave this implicit and develop only the row-sampling version of our algorithm.

---

**Algorithm 1** Optimal approximate SVD within a row subspace $\widehat{V}$.

---

EXTRACTSVD

**Input:** target matrix $A \in \mathbb{R}^{m \times n}$, subspace basis $\widehat{V} \in \mathbb{O}^{n \times k}$

**Output:** $U, \Sigma, V$, the SVD of the best approximation to $A$ within the subspace spanned by $\widehat{V}$'s columns

1. Compute $A\widehat{V}$, then $(A\widehat{V})^T A\widehat{V}$ and its SVD: $U'\Sigma'V'^T = (A\widehat{V})^T A\widehat{V}$

2. Let $V = \widehat{V}V'$, $\Sigma = (\Sigma')^{1/2}$, and $U = (A\widehat{V})V'\Sigma^{-1}$

3. Return $U, \Sigma, V$

---

The singular value decomposition is defined as follows:

**Definition 1.** *Let $A$ be an $m \times n$ real matrix of rank $\rho$. Then there exists a factorization of the form*

$$A = U\Sigma V^T , \tag{1}$$

*where $U$ and $V$ each have orthonormal columns and are of size $m \times \rho$ and $n \times \rho$, respectively, and $\Sigma$ is diagonal with entries $\sigma_1 \geq \sigma_2 \geq \ldots \geq \sigma_\rho > 0$.*

Equivalently, we can write the SVD as a weighted sum of rank-one outer products: $A = \sum_{i=1}^{\rho} \sigma_i u_i v_i^T$, where $u_i$ and $v_i$ represent the $i$th columns of $U$ and $V$. The columns $u_i$ and $v_i$ are referred to as the left and right singular vectors, while the weights $\sigma_i$ are the singular values. Though it is sometimes overkill, the SVD can be used to solve essentially any problem in numerical linear algebra. Instances of such problems abound in machine learning.

Given $m \geq n$, the exact SVD has $O(mn^2)$ runtime ($O(n^3)$ for square matrices). This is highly unscalable, rendering exact SVD impractical for large datasets. However, it is often the case that good approximations can be found using subsets of the rows or columns. Of significant interest are low-rank approximations to a matrix. The optimal $k$-rank approximation, in the sense of minimizing the squared error $||A - \widehat{A}||_F^2$, is the $k$-rank truncation of the SVD:

$$A_k = \sum_{i=1}^{k} \sigma_i u_i v_i^T = U_k \Sigma_k V_k . \tag{2}$$

$A_k$ is the projection of $A$'s rows onto the subspace spanned by the top $k$ right singular vectors, i.e., $A_k = AV_k V_k^T$. The optimality of $A_k$ implies that the columns of $V_k$ span the subspace of dimension at most $k$ in which the squared error of $A$'s row-wise projection is minimized. This leads us to a formulation of SVD approximation in which we seek to find a subspace in which $A$'s projection has sufficiently low error, then perform the SVD of $A$ in that subspace. If the subspace is substantially lower in rank/dimension than $A$, the SVD of the projection can be computed significantly faster than the SVD of the original $A$ (quadratically so, as we will have decreased the $n$ in $O(mn^2)$). An important procedure we will require is the extraction of the best approximate SVD within a given subspace $\widehat{V}$. Algorithm 1 describes this process; portions of this idea appeared in [1] and [2], but without enumeration of its properties. We state some of the key properties as a lemma.

**Lemma 1.** *Given a target matrix $A$ and a row subspace basis stored in the columns of $\widehat{V}$, EXTRACTSVD has the following properties:*

1. *Returns a full SVD, meaning $U$ and $V$ with orthonormal columns, and $\Sigma$ diagonal.*

2. *$U\Sigma V^T = A\widehat{V}\widehat{V}^T$, i.e., the extracted SVD reconstructs exactly to the projection of $A$'s rows onto the subspace spanned by $\widehat{V}$.*

3. *$U\Sigma V^T$ minimizes squared-error reconstruction of $A$ among all SVDs whose rows are restricted to the span of $\widehat{V}$.*

We omit the fairly straightforward proof. The runtime of the procedure is $O(kmn)$, where $k$ is the rank of $\widehat{V}$. As this SVD extraction will constitute the last and most expensive step of our algorithm, we therefore require a subspace discovery method that finds a subspace of sufficient quality with as low a rank $k$ as possible. This motivates the essential idea of our approach, which is to leverage the

Table 1: Distinctions between whole-matrix SVD approximation and LRMA.

| Whole-Matrix SVD Approximation | Low-Rank Matrix Approximation |
|---|---|
| True SVD: $U$, $\Sigma$, and $V$ | $\widehat{A}$ or unaligned $\widehat{V}$ & $\widehat{\Sigma}$ only |
| Addresses full-rank matrix | Fixed low-rank $k$ |
| Full-rank relative error bound | $k$-rank error bound, additive or relative |

Table 2: Distinctions between subspace construction in QUIC-SVD and previous LRMA methods.

| QUIC-SVD | Previous LRMA Methods |
|---|---|
| Iterative buildup, fast empirical error control | One-off computation, loose error bound |
| Adaptive sample size minimization | Fixed *a priori* sample size (loose) |
| Cosine tree sampling | Various sampling schemes |

geometric structure of a matrix to efficiently derive compact (i.e., minimal-rank) subspaces in which to carry out the approximate SVD.

**Previous Work**. A recent vein of work in the theory and algorithms community has focused on using sampling to solve the problem of low-rank matrix approximation (LRMA). The user specifies a desired low rank $k$, and the algorithms try to output something close to the optimal $k$-rank approximation. This problem is different from the whole-matrix SVD approximation we address, but a close relationship allow us to draw on some of the LRMA ideas. Table 1 highlights the distinctions between whole-matrix SVD approximation and LRMA. Table 2 summarizes the differences between our algorithmic approach and the more theoretically-oriented approaches taken in the LRMA work.

Each LRMA algorithm has a way of sampling to build up a subspace in which the matrix projection has bounded error. Our SVD also samples to build a subspace, so the LRMA sampling methods are directly comparable to our tree-based approach. Three main LRMA sampling techniques have emerged,[1] and we will discuss each from the perspective of iteratively sampling a row, updating a subspace so it spans the new row, and continuing until the subspace captures the input matrix to within a desired error threshold. This is how our method works, and it is similar to the framework used by Friedland et al. [1]. The key to efficiency (i.e., rank-compactness) is for each sampled row to represent well the rows that are not yet well represented in the subspace.

*Length-squared (LS) sampling*. Rows are sampled with probability proportional to their squared lengths: $p_i = ||A_{(i)}||_F^2/||A||_F^2$. LS sampling was used in the seminal work of Frieze, Kannan, and Vempala [3], and in much of the follow-on work [4, 5]. It is essentially an importance sampling scheme for the squared error objective. However, it has two important weaknesses. First, a row can have high norm while not being representative of other rows. Second, the distribution is non-adaptive, in that a point is equally likely to be drawn whether or not it is already well represented in the subspace. Both of these lead to wasted samples and needless inflation of the subspace rank.

*Residual length-squared (RLS) sampling*. Introduced by Deshpande and Vempala [2], RLS modifies the LS probabilities after each subspace update by setting $p_i = ||A_{(i)} - \Pi_V(A_{(i)})||_F^2/||A - \Pi_V(A)||_F^2$, where $\Pi_V$ represents projection onto the current subspace $V$. By adapting the LS distribution to be over residuals, this method avoids drawing samples that are already well represented in the subspace. Unfortunately, there is still nothing to enforce that any sample will be representative of other high-residual samples. Further, updating residuals requires an expensive $s$ passes through the matrix for every $s$ samples that are added, which significantly limits practical utility.

*Random projections (RP)*. Introduced by Sarlós [6], the idea is to sample linear combinations of rows, with random combination coefficients drawn from a Gaussian. This method is strong where LS and RLS are weak — because all rows influence every sample, each sample is likely to represent a sizeable number of rows. Unfortunately the combination coefficients are not informed by importance (squared length), and the sampling distribution is non-adaptive. Further, each linear combination requires a full matrix pass, again limiting practicality.

Also deserving mention is the randomized sparsification used by Achlioptas et al. [7]. Each of the LRMA sampling methods has strengths we can draw on and weaknesses we can improve upon. In particular, our cosine tree sampling method can be viewed as combining the representativeness of RP sampling with the adaptivity of RLS, which explains its empirically dominant rank efficiency.

**Algorithm 2** Cosine tree construction.

---

CTNODE
**Input:** $A \in \mathbb{R}^{m \times n}$
**Output:** cosine tree node containing the rows of $A$

     1. $N \leftarrow$ new cosine tree node

     2. $N.A \leftarrow A$

     3. $N.splitPt \leftarrow$ ROWSAMPLELS$(A)$ // split point sampled from length-squared distribution

     4. **return** $N$

CTNODESPLIT
**Input:** cosine tree node $N$
**Output:** left and right children obtained by cosine-splitting of $N$

     1. **for each** $N.A_{(i)}$, compute $c_i = |cos(N.A_{(i)}, \ N.splitPt)|$

     2. **if** $\forall i, \ c_i = 1$, **return** $nil$

     3. $c_{max} = \max\{c_i | c_i < 1\}; c_{min} = \min\{c_i\}$

     4. $A_l \leftarrow [\ ]; A_r \leftarrow [\ ]$

     5. **for** $i = 1$ to $N.nRows$

        (a) **if** $c_{max} - c_i \leq c_i - c_{min}, \ A_l \leftarrow \begin{bmatrix} A_l \\ N.A_{(i)} \end{bmatrix}$

        (b) **else** $A_r \leftarrow \begin{bmatrix} A_r \\ N.A_{(i)} \end{bmatrix}$

     6. **return** CTNODE$(A_l)$, CTNODE$(A_r)$

---

## 3 Our Approach

Rather than a fixed low-rank matrix approximation, our objective is to approximate the whole-matrix SVD with as high a rank as is required to obtain the following whole-matrix relative error bound:

$$||A - \widehat{A}||_F^2 \leq \epsilon ||A||_F^2 , \tag{3}$$

where $\widehat{A} = U\Sigma V^T$ is the matrix reconstructed by our SVD approximation. In contrast to the error bounds of previous methods, which are stated in terms of the unknown low-rank $A_k$, our error bound is in terms of the known $A$. This enables us to use a fast, empirical Monte Carlo technique to determine with high confidence when we have achieved the error target, and therefore to terminate with as few samples and as compact a subspace as possible. Minimizing subspace rank is crucial for speed, as the final SVD extraction is greatly slowed by excess rank when the input matrix is large.

We use an iterative subspace buildup as described in the previous section, with sampling governed by a new spatial partitioning structure we call the *cosine tree*. Cosine trees are designed to leverage the geometrical structure of a matrix and a partial subspace in order to quickly home in on good representative samples from the regions least well represented. Key to the efficiency of our algorithm is an efficient error checking scheme, which we accomplish by Monte Carlo error estimation at judiciously chosen stages. Such a combination of spatial partitioning trees and Monte Carlo estimation has been used before to good effect [8], and we find it to be a successful pairing here as well.

**Cosine Trees for Efficient Subspace Discovery**. The ideal subspace discovery algorithm would oracularly choose as samples the singular vectors $v_i$. Each $v_i$ is precisely the direction that, added to the subspace spanned by the previous singular vectors, will maximally decrease residual error over all rows of the matrix. This intuition is the guiding idea for cosine trees.

A cosine tree is constructed as follows. Starting with a root node, which contains all points (rows), we take its centroid as a representative to include in our subspace span, and randomly sample a point to serve as the pivot for splitting. We sample the pivot from the basic LS distribution, that being the cheapest source of information as to sample importance. The remaining points are sorted by their absolute cosines relative to the pivot point, then split according to whether they are closer to the high or low end of the cosines. The two groups are assigned to two child nodes, which are placed in a

---

**Algorithm 3** Monte Carlo estimation of the squared error of a matrix projection onto a subspace.

---

MCSqError

**Input:** $A \in \mathbb{R}^{m \times n}, \widehat{V} \in \mathbb{O}^{n \times k}, s \in \{1 \ldots m\}, \delta \in [0, 1]$

**Output:** $sqErr \in \mathbb{R}$ s.t. with probability at least $1 - \delta$, $||A - A\widehat{V}\widehat{V}^T||_F^2 \leq sqErr$

1. $S = rowSamplesLS(A, \; s)$ // sample $s$ rows from the length-squared distribution

2. **for** $i = 1$ **to** $s$ **:** // compute weighted sq. mag. of each sampled row's projection onto $V$

    (a) $wgtMagSq[i] = \frac{1}{p_{S_{(i)}}}||S_{(i)}V||_F^2$ // $p_{S_{(i)}}$ is prob. of drawing $S_i$ under LS sampling

3. $\hat{\mu} = avg(wgtMagSq); \; \hat{\sigma}^2 = var(wgtMagSq); \; magSqLB = lowBound(\hat{\mu}, \; \hat{\sigma}^2, \; s, \; \delta)$

4. **return** $||A||_F^2 - magSqLB$

---

**Algorithm 4** QUIC-SVD: fast whole-matrix approximate SVD with relative error control.

---

QUIC-SVD

**Input:** $A \in \mathbb{R}^{m \times n}, \epsilon \in [0, 1]$, and $\delta \in [0, 1]$

**Output:** an SVD $U, \Sigma, V$ s.t. $\widehat{A} = U\Sigma V^T$ satisfies $||A - \widehat{A}||_F^2 \leq \epsilon||A||_F^2$ with probability at least $1 - \delta$

1. $V = [ \; ]; mcSqErr = ||A||_F^2; N_{root} = \text{CTNode}(A)$

2. $Q = \text{EmptyPriorityQueue}(); \; Q.insert(N_{root}, \; 0)$

3. **do until** $mcSqErr \leq \epsilon||A||_F^2$**:**

    (a) $N = Q.pop(); C = \text{CTNodeSplit}(N)$ // $C = \{N_l, N_r\}$, the children of $N$

    (b) Remove $N$'s contributed basis vector from $V$

    (c) **for each** $N_c \in C$ **:**

      i. $V = [V \; \text{MGS}(V, \; N_c.centroid)]$ // MGS = modified Gram-Schmidt orthonormalization

    (d) **for each** $N_c \in C$ **:**

      i. $errC = \text{MCSqError}(N_c.A, \; V, \; O(\log[N_c.nRows]), \; \delta)$

      ii. $Q.insert(N_c, \; errC)$

    (e) $mcSqErr = \text{MCSqError}(A, \; V, \; O(\log m), \; \delta)$

4. **return** $\text{ExtractSVD}(A, \; V)$

---

queue prioritized by the residual error of each node. The process is then repeated according to the priority order of the queue. Algorithm 2 defines the splitting process.

Why do cosine trees improve sampling efficiency? By prioritizing expansion by the residual error of the frontier nodes, sampling is always focused on the areas with maximum potential for error reduction. Since cosine-based splitting guides the nodes toward groupings with higher parallelism, the residual magnitude of each node is increasingly likely to be well captured along the direction of the node centroid. Expanding the subspace in the direction of the highest-priority node centroid is therefore a good guess as to the direction that will maximally reduce residual error. Thus, cosine tree sampling approximates the ideal of oracularly sampling the true singular vectors.

### 3.1 QUIC-SVD

**Strong error control**. Algorithm 4, QUIC-SVD (QUantized Iterative Cosine tree)[2], specifies a way to leverage cosine trees in the construction of an approximate SVD while providing a strong probabilistic error guarantee. The algorithm builds a subspace by expanding a cosine tree as described above, checking residual error after each expansion. Once the residual error is sufficiently low, we return the SVD of the projection into the subspace. Note that exact error checking would require an expensive $O(k^2mn)$ total cost, where $k$ is the final subspace rank, so we instead use a Monte Carlo error estimate as specified in Algorithm 3. We also employ Algorithm 3 for the error estimates used in node prioritization. With Monte Carlo instead of exact error computations, the total cost for error checking decreases to $O(k^2n \log m)$, a significant practical reduction.

The other main contributions to runtime are: 1) $k$ cosine tree node splits for a total of $O(kmn)$, 2) $O(k)$ single-vector Gram-Schmidt orthonormalizations at $O(km)$ each for a total of $O(k^2m)$, and 3) final SVD extraction at $O(kmn)$. Total runtime is therefore $O(kmn)$, with the final projection onto the subspace being the costliest step since the $O(kmn)$ from node splitting is a very loose worst-case bound. We now state the QUIC-SVD error guarantee.

**Theorem 1.** *Given a matrix $A \in \mathbb{R}^{m \times n}$ and $\epsilon, \delta \in [0, 1]$, the algorithm* QUIC-SVD *returns an SVD $U, \Sigma, V$ such that $\widehat{A} = U\Sigma V^T$ satisfies $||A - \widehat{A}||_F^2 \leq \epsilon ||A||_F^2$ with probability at least $1 - \delta$.*

*Proof sketch.* The algorithm terminates after $mcSqErr \leq \epsilon ||A||_F^2$ with a call to EXTRACTSVD. From Lemma 1 we know that EXTRACTSVD returns an SVD that reconstructs to $A$'s projection onto $V$ (i.e., $\widehat{A} = AVV^T$). Thus, we have only to show that $mcSqErr$ in the terminal iteration is an upper bound on the error $||A - \widehat{A}||_F^2$ with probability at least $1 - \delta$. Note that intermediate error checks do not affect the success probability, since they only ever tell us to continue expanding the subspace, which is never a failure. From the Pythagorean theorem, $||A - AVV^T||_F^2 = ||A||_F^2 - ||AVV^T||_F^2$, and, since rotations do not affect lengths, $||AVV^T||_F^2 = ||AV||_F^2$. The call to MCSQERROR (step 3(e)) performs a Monte Carlo estimate of $||AV||_F^2$ in order to estimate $||A||_F^2 - ||AV||_F^2$. It is easily verified that the length-squared-weighted sample mean used by MCSQERROR produces an unbiased estimate of $||AV||_F^2$. By using a valid confidence interval to generate a $1 - \delta$ lower bound on $||AV||_F^2$ from the sample mean and variance (e.g., Theorem 1 of [9] or similar), MCSQERROR is guaranteed to return an upper bound on $||A||_F^2 - ||AV||_F^2$ with probability at least $1 - \delta$, which establishes the theorem. $\qquad\square$

**Relaxed error control**. Though the QUIC-SVD procedure specified in Algorithm 4 provides a strong error guarantee, in practice its error checking routine is overconservative and is invoked more frequently than necessary. For practical usage, we therefore approximate the strict error checking of Algorithm 4 by making three modifications:

1. Set $mcSqErr$ to the mean, rather than the lower bound, of the MCSQERROR estimate.
2. At each error check, estimate $mcSqErr$ with several repeated Monte Carlo evaluations (i.e., calls to MCSQERROR), terminating only if they all result in $mcSqErr \leq \epsilon ||A||_F^2$.
3. In each iteration, use a linear extrapolation from past decreases in error to estimate the number of additional node splits required to achieve the error target. Perform this projected number of splits before checking error again, thus eliminating needless intermediate error checks.

Although these modifications forfeit the strict guarantee of Theorem 1, they are principled approximations that more aggressively accelerate the computation while still keeping error well under control (this will be demonstrated empirically). Changes 1 and 2 are based on the fact that, because $mcSqErr$ is an unbiased estimate generated by a sample mean, it obeys the Central Limit Theorem and thus approaches a normal distribution centered on the true squared error. Under such a symmetric distribution, the probability that a single evaluation of $mcSqErr$ will exceed the true error is 0.5. The probability that, in a series of $x$ evaluations, at least *one* of them will exceed the true error is approximately $1 - 0.5^x$ (1 minus the probability that they all come in below the true error). The probability that at least one of our $mcSqErr$ evaluations results in an upper bound on the true error (i.e., the probability that our error check is correct) thus goes quickly to 1. In our experiments, we use $x = 3$, corresponding to a success probability of approximately 0.9 (i.e., $\delta \approx 0.1$).

Change 3 exploits that fact that the rate at which error decreases is typically monotonically non-increasing. Thus, extrapolating the rate of error decrease from past error evaluations yields a conservative estimate of the number of splits required to achieve the error target. Naturally, we have to impose limits to guard against outlier cases where the estimated number is unreasonably high. Our experiments limit the size of the split jumps to be no more than 100.

## 4 Performance

We report the results of two sets of experiments, one comparing the sample efficiency of cosine trees to previous LRMA sampling methods, and the other evaluating the composite speed and error performance of QUIC-SVD. Due to space considerations we give results for only two datasets, and

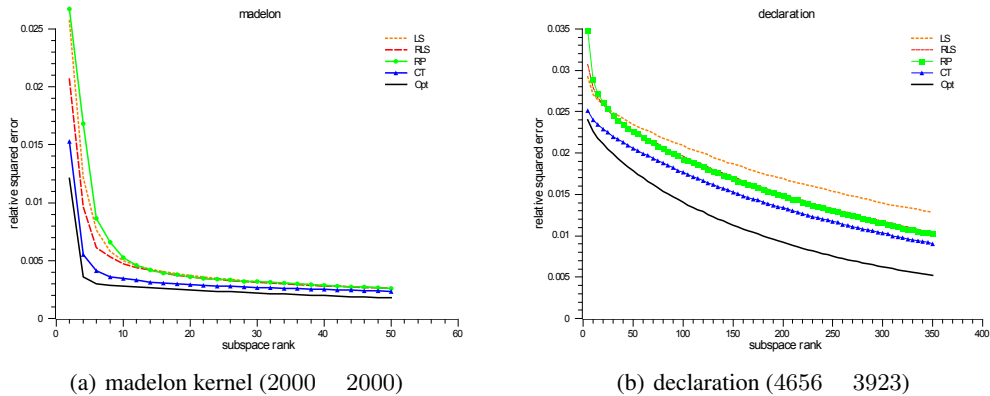

Figure 1: Relative squared error vs. subspace rank for various subspace discovery methods. LS is length-squared, RLS is residual length-squared, RP is random projection, and CT is cosine tree.

due to the need to compute the exact SVD as a baseline we limit ourselves to medium-sized matrices. Nonetheless, these results are illustrative of the more general performance of the algorithm.

**Sample efficiency**. Because the runtime of our algorithm is $O(kmn)$, where $k$ is the final dimension of the projection subspace, it is critical that we use a sampling method that achieves the error target with the minimum possible subspace rank $k$. We therefore compare our cosine tree sampling method to the previous sampling methods proposed in the LRMA literature. Figure 1 shows results for the various sampling methods on two matrices, one a $2000 \times 2000$ Gaussian kernel matrix produced by the Madelon dataset from the NIPS 2003 Workshop on Feature Extraction (madelon kernel), and the other a $4656 \times 3923$ scan of the US Declaration of Independence (declaration). Plotted is the relative squared error of the input matrix's projection onto the subspaces generated by each method at each subspace rank. Also shown is the optimal error produced by the exact SVD at each rank.

Both graphs show cosine trees dominating the other methods in terms of rank efficiency. This dominance has been confirmed by many other empirical results we lack space to report here. It is particularly interesting how closely the cosine tree error can track that of the exact SVD. This would seem to give some justification to the principle of grouping points according to their degree of mutual parallelism, and validates our use of cosine trees as the sampling mechanism for QUIC-SVD.

**Speedup and error.** In the second set of experiments we evaluate the runtime and error performance of QUIC-SVD. Figure 2 shows results for the madelon kernel and declaration matrices. On the top row we show how speedup over exact SVD varies with the target error $\epsilon$. Speedups range from 831 at $\epsilon = 0.0025$ to over 3,600 at $\epsilon = 0.023$ for madelon kernel, and from 118 at $\epsilon = 0.01$ to nearly 20,000 at $\epsilon = 0.03$ for declaration. On the bottom row we show the actual error of the algorithm in comparison to the target error. While the actual error is most often slightly above the target, it nevertheless hugs the target line quite closely, never exceeding the target by more than 10%. Overall, the several-order-of-magnitude speedups and controlled error shown by QUIC-SVD would seem to make it an attractive option for any algorithm computing costly SVDs.

## 5   Conclusion

We have presented a fast approximate SVD algorithm, QUIC-SVD, and demonstrated several-order-of-magnitude speedups with controlled error on medium-sized datasets. This algorithm differs from previous related work in that it addresses the whole-matrix SVD, not low-rank matrix approximation, it uses a new efficient sampling procedure based on cosine trees, and it uses empirical Monte Carlo error estimates to adaptively minimize needed sample sizes, rather than fixing a loose sample size *a priori*. In addition to theoretical justifications, the empirical performance of QUIC-SVD argues for its effectiveness and utility. We note that a refined version of QUIC-SVD is forthcoming. The new version is greatly simplified, and features even greater speed with a deterministic error guarantee. More work is needed to explore the SVD-using methods to which QUIC-SVD can be applied, particularly with an eye to how the introduction of controlled error in the SVD will

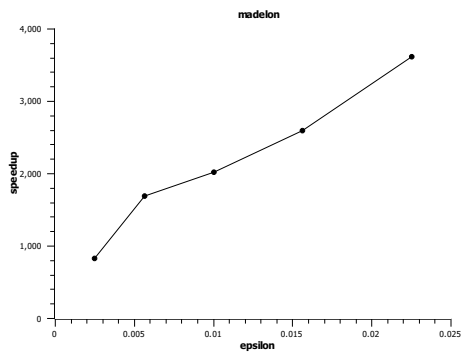

(a) speedup - madelon kernel

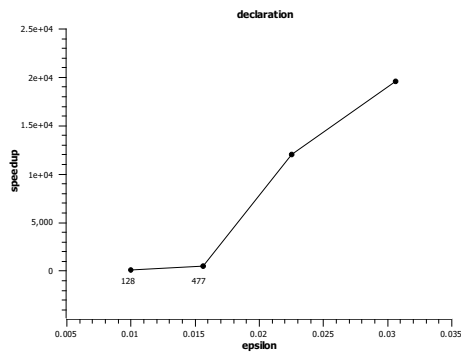

(b) speedup - declaration

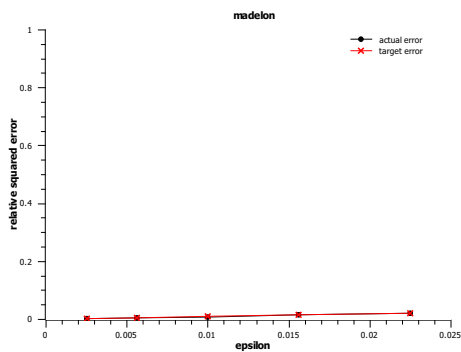

(c) relative error - madelon kernel

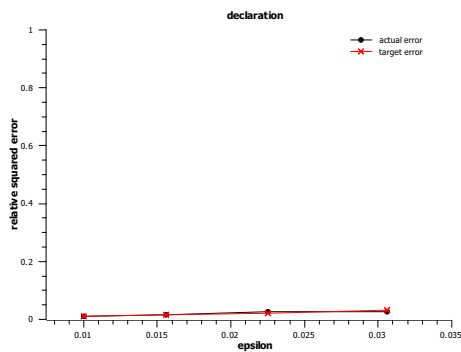

(d) relative error - declaration

Figure 2: Speedup and actual relative error vs.    for QUIC-SVD on madelon kernel and declaration.

affect the quality of the methods using it. We expect there will be many opportunities to enable new applications through the scalability of this approximation.

## Footnotes

[1]Note that our summary of related work is necessarily incomplete due to space constraints; our intent is to summarize the essential results from the LRMA literature inasmuch as they pertain to our approach.

[2]Quantized alludes to each node being represented by a single point that is added to the subspace basis.

# References

[1] S. Friedland, A. Niknejad, M. Kaveh, and H. Zare. Fast Monte-Carlo Low Rank Approximations for Matrices. In *Proceedings of Int. Conf. on System of Systems Engineering*, 2006.

[2] A. Deshpande and S. Vempala. Adaptive Sampling and Fast Low-Rank Matrix Approximation. In *10th International Workshop on Randomization and Computation (RANDOM06)*, 2006.

[3] A. M. Frieze, R. Kannan, and S. Vempala. Fast Monte-Carlo Algorithms for Finding Low-Rank Approximations. In *IEEE Symposium on Foundations of Computer Science*, pages 370–378, 1998.

[4] P. Drineas, R. Kannan, and M. W. Mahoney. Fast Monte Carlo Algorithms for Matrices II: Computing a Low-Rank Approximation to a Matrix. *SIAM Journal on Computing*, 36(1):158–183, 2006.

[5] P. Drineas, E. Drinea, and P. S. Huggins. An Experimental Evaluation of a Monte-Carlo Algorithm for Singular Value Decomposition. *Lectures Notes in Computer Science*, 2563:279–296, 2003.

[6] T. Sarlos. Improved Approximation Algorithms for Large Matrices via Random Projections. In *47th IEEE Symposium on Foundations of Computer Science (FOCS)*, pages 143–152, 2006.

[7] D. Achlioptas, F. McSherry, and B. Scholkopf. Sampling Techniques for Kernel Methods. In *Advances in Neural Information Processing Systems (NIPS) 17*, 2002.

[8] M. P. Holmes, A. G. Gray, and C. L.Isbell, Jr. Ultrafast Monte Carlo for Kernel Estimators and Generalized Statistical Summations. In *Advances in Neural Information Processing Systems (NIPS) 21*, 2008.

[9] J. Audibert, R. Munos, and C. Szepesvari. Variance estimates and exploration function in multi-armed bandits. Technical report, CERTIS, 2007.

